# Cross-Validation Estimates IMSE

**Mark Plutowski** †⋆        **Shinichi Sakata** ‡        **Halbert White** ‡⋆

† Department of Computer Science and Engineering
‡ Department of Economics
⋆ Institute for Neural Computation
University of California, San Diego

## Abstract

Integrated Mean Squared Error (IMSE) is a version of the usual mean squared error criterion, averaged over all possible training sets of a given size. If it could be observed, it could be used to determine optimal network complexity or optimal data subsets for efficient training. We show that two common methods of cross-validating average squared error deliver unbiased estimates of IMSE, converging to IMSE with probability one. These estimates thus make possible approximate IMSE-based choice of network complexity. We also show that two variants of cross validation measure provide unbiased IMSE-based estimates potentially useful for selecting optimal data subsets.

## 1 Summary

To begin, assume we are given a fixed network architecture. (We dispense with this assumption later.) Let $z^N$ denote a given set of $N$ training examples. Let $Q_N(z^N)$ denote the expected squared error (the expectation taken over all possible examples) of the network after being trained on $z^N$. This measures the quality of fit afforded by training on a given set of $N$ examples.

Let $IMSE_N$ denote the Integrated Mean Squared Error for training sets of size $N$. Given reasonable assumptions, it is straightforward to show that $IMSE_N = E[Q_N(Z^N)] - \sigma^2$, where the expectation is now over all training sets of size $N$, $Z^N$ is a random training set of size $N$, and $\sigma^2$ is the noise variance.

Let $C_N = C_N(z^N)$ denote the "delete-one cross-validation" squared error measure for a network trained on $z^N$. $C_N$ is obtained by training networks on each of the $N$ training sets of size $N-1$ obtained by deleting a single example; the measure follows

by computing squared error for the corresponding deleted example and averaging the results. Let $G_{N,M} = G_{N,M}(z^N, \tilde{z}^M)$ denote the "generalization" measure obtained by separating the available data of size $N + M$ into a training set $z^N$ of size $N$, and a validation ("test") set $\tilde{z}^M$ of size $M$; the measure follows by training on $z^N$ and computing averaged squared error over $\tilde{z}^M$.

We show that $C_N$ is an unbiased estimator of $E[Q_{N-1}(Z^N)]$, and hence, estimates $IMSE_{N-1}$ up to noise variance. Similarly, $G_{N,M}$ is an unbiased estimator of $E[Q_N(Z^N, \tilde{Z}^M)]$. Given reasonable conditions on the estimator and on the data generating process we demonstrate convergence with probability 1 of $G_{N,M}$ and $C_N$ to $E[Q_N(Z^N)]$ as $N$ and $M$ grow large.

A direct consequence of these results is that when choice is restricted to a set of network architectures whose complexity is bounded above a priori, then choosing the architecture for which either $C_N$ (or $G_{N,M}$) is minimized leads to choice of the network for which $IMSE_N$ is nearly minimized for all $N$ (respectively, $N, M$) sufficiently large.

We also provide results for training sets sampled at particular inputs. Conditional $IMSE$ is an appealing criterion for evaluating a particular choice of training set in the presence of noise. These results demonstrate that delete-one cross-validation estimates average MSE (the average taken over the given set of inputs,) and that hold-out set cross-validation gives an unbiased estimate of $E[Q_N(Z^N)|Z^N = (x^N, Y^N)]$, given a set of $N$ input values $x^N$ for which corresponding (random) output values $Y^N$ are obtained. Either cross-validation measure can therefore be used to select a representative subset of the entire dataset that can be used for data compaction, or for more efficient training (as training can be faster on smaller datasets) [4].

## 2  Definitions
### 2.1  Learning Task
We consider the learning task of determining the relationship between a random vector $X$ and a random scalar $Y$, where $X$ takes values in a subset $\mathbf{X}$ of $\Re^r$, and $Y$ takes values in a subset $\mathbf{Y}$ of $\Re$. (e.g. $\mathbf{X} = \Re^r$ and $\mathbf{Y} = \Re$). We refer to $\mathbf{X}$ as the *input space*. The learning task is thus one of training a neural network with $r$ inputs and one output. It is straightforward to extend the following analysis to networks with multiple targets. We make the following assumption on the observations to be used in the training of the networks.

**Assumption 1** $\mathbf{X}$ *is a Borel subset of* $\Re^r$ *and* $\mathbf{Y}$ *is a Borel subset of* $\Re$. *Let* $\mathbf{Z} \equiv \mathbf{X} \times \mathbf{Y}$ *and* $\Omega \equiv \mathbf{Z}^\infty \equiv \times_{i=1}^\infty Z$. *Let* $(\Omega, \mathcal{F}, \mathcal{P})$ *be a probability space with* $\mathcal{F} = \mathcal{B}(\Omega)$. *The observations on* $Z \equiv (X', Y)'$ *to be used in the training of the network are a realization of an i.i.d. stochastic process* $\{Z_i \equiv (X_i', Y_i)' : \Omega \to \mathbf{X} \times \mathbf{Y}\}$.

When $\omega \in \Omega$ is fixed, we write $z_i \equiv Z_i(\omega)$ for each $i = 1, 2, \ldots$. Also write $Z^N \equiv (Z_1, \ldots, Z_N)$ and $z^n \equiv (z_1, \ldots, z_n)$.

Assumption 1 allows uncertainty caused by measurement errors of observations as well as a *probabilistic* relationship between $X$ and $Y$. It, however, does not prevent a deterministic relation ship between $X$ and $Y$ such that $Y = g(X)$ for some measurable mapping $g : \Re^r \to \Re$.

We suppose interest attaches to the conditional expectation of $Y$ given $X$, written $g(x) = E(Y|X)$. The next assumption guarantees the existence of $E(Y_i|X_i)$ and $E(\varepsilon_i|X_i)$, $\varepsilon_i \equiv Y_i - E(Y_i|X_i)$. Next, for convenience, we assume homoscedasticity of the conditional variance of $Y_i$ given $X_i$.

**Assumption 2** $E(Y^2) < \infty$.

**Assumption 3** $E(\varepsilon_1^2|X_1) = \sigma^2$, where $\sigma^2$ is a strictly positive constant.

## 2.2   Network Model

Let $f^p(\cdot, \cdot) : \mathbf{X} \times \mathbf{W}^p \to \mathbf{Y}$ be a network function with the "weight space" $\mathbf{W}^p$, where $p$ denotes the dimension of the "weight space" (the number of weights.) We impose some mild conditions on the network architecture.

**Assumption 4** *For each $p \in \{1, 2, \ldots, \bar{p}\}$, $\bar{p} \in \mathbf{N}$, $\mathbf{W}^p$ is a compact subset of $\Re^p$, and $f^p : \mathbf{X} \times \mathbf{W}^p \to \Re$ satisfies the following conditions:*

  *1. $f^p(\cdot, w) : \mathbf{X} \to \mathbf{Y}$ is measurable for each $w \in \mathbf{W}^p$;*

  *2. $f^p(x, \cdot) : \mathbf{W}^p \to \mathbf{Y}$ is continuous for all $x \in \mathbf{X}$.*

We further make a joint assumption on the underlying data generating process and the network architecture to assure that the training dataset and the networks behaves appropriately.

**Assumption 5** *There exists a function $D : \mathbf{X} \to \Re_+ \equiv [0, \infty)$ such that for each $x \in \mathbf{X}$ and $w \in \mathbf{W}^p$, $|f^p(x, w)| \le D(x)$, and $\mathrm{E}\left[(D(X))^2\right] < \infty$.*

Hence, $f^p$ is square integrable for each $w^p \in \mathbf{W}^p$. We will measure network performance using mean squared error, which for weights $w^p$ is given by $\lambda(w^p; p) \equiv E\left[(Y - f^p(X, w^p))^2\right]$. The optimal weights are the weights that minimize $\lambda(w^p; p)$. The set of all optimal weights are given by $\mathbf{W}^{p*} \equiv \{w^* \in \mathbf{W}^p : \lambda(w^*; p) \le \lambda(w; p)$ for any $w \in \mathbf{W}^p\}$. The index of the best network is $p^*$, given by the smallest $p$ minimizing $\min_{w^p \in \mathbf{W}^p} \lambda(w^p; p)$, $p \in \{1, 2, \ldots, \bar{p}\}$.

## 2.3   Least-Squares Estimator

When assumptions 1 and 4 hold, the nonlinear least-squares estimator exists. Formally, we have

**Lemma 1** *Suppose that Assumptions 1 and 4 hold. Then 1. For each $N \in \mathbf{N}$, there exists a measurable function $l_N(\cdot; p) : \mathbf{Z}^N \to \mathbf{W}^p$ such that $l_N(Z^N; p)$ solves the following problem with probability one: $\min_{w \in \mathbf{W}^p} N^{-1} \sum_{i=1}^{N} (Y_i - f(X_i, w))^2$. 2. $\lambda(\cdot; p) : \mathbf{W}^p \to \Re$ is continuous on $\mathbf{W}^p$, and $\mathbf{W}^{p*}$ is not empty.*

For convenience, we also define $\hat{w}_N^p : \Omega \to \mathbf{W}^p$ by $\hat{w}_N^p(\omega) \equiv l_N(Z^N(\omega); p)$ for each $\omega \in \Omega$. Next let $i_1, i_2, \ldots, i_N$ be distinct natural numbers and let $\check{Z}^N = (Z_{i_1}, \ldots, Z_{i_N})'$. Then $l_N(\check{Z}^N)$ given above solves $\frac{1}{N} \sum_{j=1}^{N} (Y_{i_j} - f(X_{i_j}, w^p))^2$ with probability one. In particular, we will consider the estimate using the dataset $z_{-i}^N$ made by deleting the $i$th observation from $z^N$. Let $Z_{-i}^N$ be a random matrix made

by deleting the $i$th row from $Z^N$. Thus, $l_{N-1}(Z^N_{-i};p)$ is a measurable least squares estimator and we can consider its probabilistic behavior.

## 3    Integrated Mean Squared Error

Integrated Mean Squared Error (IMSE) has been used to regulate network complexity [9]. Another (conditional) version of IMSE is used as a criterion for evaluating training examples [5, 6, 7, 8]. The first version depends only on the sample size, not the particular sample. The second (conditional) version depends additionally upon the observed location of the examples in the input space.

### 3.1    Unconditional IMSE

The (unconditional) mean squared error (MSE) of the network output at a particular input value $\check{x}$ is

$$M_N(\check{x};p) = E\left[\{g(\check{x}) - f(\check{x}, l_N(Z^N;p))\}^2\right]. \tag{1}$$

Integrating MSE over all possible inputs gives the unconditional IMSE:

$$IMSE_N(p) = \int [M_N(x,;p)]\,\mu(dx) \tag{2}$$

$$= E[M_N(X;p)], \tag{3}$$

where $\mu$ is the input distribution.

### 3.2    Conditional IMSE

To evaluate exemplars obtained at inputs $x^N$, we modify Equation (1) by conditioning on $x^N$, giving

$$M_N(\check{x}|x^N;p) = E\left[\{g(\check{x}) - f(\check{x}, l_N(Z^N))\}^2|X^N = x^N\right].$$

The conditional IMSE (given inputs $x^N$) is then

$$IMSE_N(x^N;p) = \int M_N(x|x^N;p)\mu(dx) \tag{4}$$

$$= E[M_N(X|x^N;p)]. \tag{5}$$

## 4    Cross-Validation

Cross-validatory measures have been used successfully to assess the performance of a wide range of estimators [10, 11, 12, 13, 14, 15]. Cross-validatory measures have been derived for various performance criteria, including the Kullback-Liebler Information Criterion (KLIC) and the Integrated Squared Error (ISE, asymptotically equivalent to IMSE) [16]. Although provably inappropriate in certain applications [17, 18], optimality and consistency results for the cross-validatory measures have been obtained for several estimators, including linear regression, orthogonal series, splines, histograms, and kernel density estimators [16, 19, 20, 21, 22, 23, 24]. The authors are not aware of similar results applicable to neural networks, although two more general, but weaker results do apply [26]. A general result applicable to neural networks shows asymptotic equivalence between cross-validation and Akaike's Criterion for network selection [25, 29], as well as between cross-validation and Moody's Criterion [30, 29].

## 4.1  Expected Network Error

Given our assumptions, we can relate cross-validation to IMSE. For clarity and notational convenience, we first introduce a measure of network error closely related to IMSE. For each weight $w^p \in \mathbf{W}^p$, we have defined the mean squared error $\lambda(w^p; p)$ in Section 2.2. We define $Q_N$ to map each dataset to the mean squared error of the estimated network

$$Q_N(z^N; p) \equiv \lambda(l_N(z^N; p); p).$$

When Assumption 3 holds, we have

$$\begin{aligned}
\lambda(w^p; p) &= E\left[(g(X) - f(X, w^p))^2\right] + \sigma^2 \\
&= E\left[(g(X_{N+1}) - f(X_{N+1}, w^p))^2\right] + \sigma^2
\end{aligned}$$

as is easily verified. We therefore have

$$Q_N(z^N; p) = E\left[(g(X_{N+1}) - f(X_{N+1}, l_N(Z^N; p)))^2 | Z^N = z^N\right] + \sigma^2.$$

Thus, by using the law of iterated expectations, we have

$$E\left[Q_N(Z^N; p)\right] = IMSE_N(p) + \sigma^2.$$

Likewise, given $x^N \in \mathbf{X}^N$,

$$E[Q_N(Z^N; p)|X^N = x^N] = IMSE(x^N; p) + \sigma^2. \tag{6}$$

## 4.2  Cross-Validatory Estimation of Error

In practice we work with observable quantities only. In particular, we must estimate the error of network $p$ over novel data ("generalization") from a finite set of examples. Such an estimate is given by the delete-one cross-validation measure:

$$C_N(z^N; p) = \frac{1}{N} \sum_{i=1}^{N} \left(y_i - f(x_i, l_{N-1}(z_{-i}^N; p))\right)^2 \tag{7}$$

where $z_{-i}^N$ denotes the training set obtained by deleting the $i$th example. Using $z_{-i}^N$ instead of $z^N$ avoids a downward bias due to testing upon examples used in training, as we show below (Theorem 3.) Another version of cross-validation is commonly used for evaluating "generalization" when an abundant supply of novel data is available for use as a "hold-out" set:

$$G_{N,M}(z^N, \tilde{z}^M; p) = \frac{1}{M} \sum_{i=1}^{M} \left(\tilde{y}_i - f(\tilde{x}_i, l_N(z^N; p))\right)^2, \tag{8}$$

where $\tilde{z}^M = (z_{N+1}, \ldots, z_{N+M})$.

## 5  Expectation of the Cross-Validation Measures

We now consider the relation between cross-validation measure and IMSE. We examine delete-one cross-validation first.

**Proposition 1 (Unbiasedness of $C_N$)** *Let Assumptions 1 through 5 hold. Then for given $N$, $C_N$ is an unbiased estimator of $IMSE_{N-1}(p) + \sigma^2$:*

$$E\left[C_N(Z^N; p)\right] = IMSE_{N-1}(p) + \sigma^2. \tag{9}$$

With hold-out set cross-validation, the validation set $Z^M$ gives i.i.d. information regarding points outside of the training set $Z^N$.

**Proposition 2 (Unbiasedness of $G_{N,M}$)** *Let Assumptions 1 through 5 hold. Let $\tilde{Z}^M \equiv (Z_{N+1}, \dots, Z_M)'$. Then for given $N$ and $M$, $G_{N,M}$ is an unbiased estimator of $IMSE_N(p) + \sigma^2$:*

$$E\left[G_{N,M}(Z^N, \tilde{Z}^M; p)\right] = IMSE_N(p) + \sigma^2. \tag{10}$$

The latter result is appealing for large $M, N$. We expect delete-one cross-validation to be more appealing when training data is not abundant.

## 6   Expectation of Cross-Validation when Sampling at Selected Inputs

We obtain analogous results for training sets obtained by sampling at a given set of inputs $x^N$. We first consider the result for delete-one cross-validation.

**Proposition 3 (Expectation of $C_N$ given $x^N$)** *Let Assumptions 1 through 5 hold. Then, given a particular set of inputs, $x^N$, $C_N$ is an unbiased estimator of average $MSE_{N-1} + \sigma^2$, the average taken over $x^N$:*

$$E\left[C_N(Z^N, p)\middle| X^N = x^N\right] \;=\; \frac{1}{N}\sum_{i=1}^{N} M_{N-1}(x_i | x_{-i}^N; p) + \sigma^2,$$

*where $x_{-i}^N$ is a matrix made by deleting the ith row of $x^N$.*

This essentially gives an estimate of $MSE_{N-1}$ limited to $x \in x^N$, losing a degree of freedom while providing no estimate of the $MSE$ off of the training points. For this average to converge to $IMSE_{N-1}$, it will suffice for the empirical distribution of $x^N$, $\bar{\mu}^N$, to converge to $\mu_N$, i.e., $\bar{\mu}_N \Rightarrow \mu_N$. We obtain a stronger result for hold-out set cross-validation. The hold-out set gives independent information on $MSE_N$ off of the training points, resulting in an estimate of $IMSE_N$ for given $x^N$.

**Proposition 4 (Expectation of $G_{N,M}$ given $x^N$)** *Let Assumptions 1 through 5 hold. Let $\tilde{Z}^M \equiv (Z_{N+1}, \dots, Z_{N+M})'$. Then, given a particular set of inputs, $x^N$, $G_{N,M}$ is an unbiased estimator of of $IMSE_N(x^N; p) + \sigma^2$:*

$$E\left[G_{N,M}(Z^N, \tilde{Z}^M; p)\middle| X^N = x^N\right] \;=\; IMSE_N(x^N; p) + \sigma^2.$$

## 7   Strong Convergence of Hold-Out Set Cross-Validation

Our conditions deliver not only unbiasedness, but also convergence of hold-out set cross-validation to $IMSE_N$, with probability 1.

**Theorem 1 (Convergence of Hold-Out Set w.p. 1)** *Let                Assumptions 1 through 5 hold. Also let $\tilde{Z}^M \equiv (Z_{N+1}, \dots, Z_{N+M})'$. If for some $A > 0$ a sequence $\{M_N\}$ of natural numbers satisfies $M_N > AN$ for any $N = 1, 2, \dots$, then*

$$G_{N,M_N}(Z^N, \tilde{Z}^{M_N}; p) - E[Q_N(Z^N; p)] \xrightarrow{a.s.} 0, \text{ as } N \to \infty.$$

# 8    Strong Convergence of Delete-One Cross-Validation

Given an additional condition (uniqueness of optimal weights) we can show strong convergence for delete-one cross-validation. First we establish uniform convergence of the estimators $\hat{w}^p(Z^N_{-i})$ to optimal weights (uniformly over $1 \leq i \leq N$.)

**Theorem 2** *Let Assumptions 1 through 5 hold. Let $Z^N_{-k}$ be the dataset made by deleting the kth observation from $Z^N$. Then*

$$\max_{1 \leq i \leq N} d\left(l_{N-1}(Z^N_{-i}; p), \mathbf{W}^{p*}\right) \to 0 \ a.s.\text{-}\mathcal{P} \ as \ N \to \infty, \tag{11}$$

*where $d(w, \mathbf{W}^{p*}) \equiv \inf_{w^* \in \mathbf{W}^{p*}} \|w - w^*\|$.*

This convergence result leads to the next result that the delete-one cross validation measure does not under-estimate the optimized MSE, namely, $\inf_{w^p \in \mathbf{W}^p} \lambda(w^p; p)$.

**Theorem 3** *Let Assumptions 1 through 5 hold. Then*

$$\liminf_{N \to \infty} C_N(Z^N; p) \geq \min_{w \in \mathbf{W}^p} \lambda(w; p) \ a.s.\text{-}\mathcal{P}.$$

When the optimum weight is unique, we have a stronger result about convergence of the delete-one cross validation measure.

**Assumption 6** $\mathbf{W}^{p*}$ *is a singleton, i.e., $\mathbf{W}^{p*}$ has only one element.*

**Theorem 4** *Let Assumptions 1 through 6 hold. Then*

$$C_N(Z^N; p) - E\left[Q_N(Z^N; p)\right] \to 0 \ a.s. \ as \ N \to \infty.$$

# 9    Conclusion

Our results justify the intuition that cross-validation measures unbiasedly and consistently estimate the expected squared error of networks trained on finite training sets, therefore providing means of obtaining $IMSE$-approximate methods of selecting appropriate network architectures, or for evaluating particular choice of training set.

Use of these cross-validation measures therefore permits us to avoid underfitting the data, asymptotically. Note, however, that although we also thereby avoid overfitting asymptotically, this avoidance is not necessarily accomplished by choosing a minimally complex architecture. The possibility exists that $IMSE_{N-1}(p) = IMSE_{N-1}(p+1)$. Because our cross-validated estimates of these quantities are random we may by chance observe $C_N(Z^N; p) > C_N(Z^N; p+1)$ and therefore select the more complex network, even though the less complex network is equally good. Of course, because the IMSE's are the same, no performance degradation (overfitting) will result in this solution.

**Acknowledgements**

This work was supported by NSF grant IRI 92-03532. We thank David Wolpert, Jan Larsen, Jeff Racine, Vjachislav Krushkal, and Patrick Fitzsimmons for valuable discussions.

# References

[1] White, H. 1989. "Learning in Artificial Neural Networks: A Statistical Perspective." Neural Computation, 1 4, pp.425-464. MIT Press, Cambridge, MA.

[2] Plutowski, Mark E., Shinichi Sakata, and Halbert White. 1993. "Cross-validation delivers strongly consistent unbiased estimates of Integrated Mean Squared Error." To appear.

[3] Plutowski, Mark E., and Halbert White. 1993. "Selecting concise training sets from clean data." IEEE Transactions on Neural Networks. 4, 3, pp.305-318.

[4] Plutowski, Mark E., Garrison Cottrell, and Halbert White. 1992. "Learning Mackey-Glass from 25 examples, Plus or Minus 2." (Presented at 1992 Neural Information Processing Systems conference.) Jack D. Cowan, Gerald Tesauro, Joshua Aspector (eds.), Advances in neural information processing systems 6, San Mateo, CA: Morgan Kaufmann Publishers.

[5] Fedorov, V.V. 1972. Theory of Optimal Experiments. Academic Press, New York.

[6] Box,G., and N.Draper. 1987. Empirical Model-Building and Response Surfaces. Wiley, New York.

[7] Khuri, A.I., and J.A.Cornell. 1987. Response Surfaces (Designs and Analyses). Marcel Dekker, Inc., New York.

[8] Faraway, Julian J. 1990. "Sequential design for the nonparametric regression of curves and surfaces." Technical Report #177, Department of Statistics, The University of Michigan.

[9] Geman, Stuart, Elie Bienenstock, and René Doursat. 1992. "Neural networks and the bias/variance dilemma." Neural Computation. 4, 1, 1-58.

[10] Stone, M. 1959. "Application of a measure of information to the design and comparison of regression experiments." Annals Math. Stat. 30 55-69

[11] "Cross-validatory choice and assessment of statistical predictions." J.R. Statist. Soc. B. 36, 2, 111-47.

[12] Bowman, Adrian W. 1984. "An alternative method of cross-validation for the smoothing of density estimates." Biometrika (1984), 71, 2, pp. 353-60.

[13] Bowman, Adrian W., Peter Hall, D.M. Titterington. 1984. "Cross-validation in nonparametric estimation of probabilities and probability densities." Biometrika (1984), 71, 2, pp. 341-51.

[14] Marron, M. 1987. "A comparison of cross-validation techniques in density estimation." The Annals of Statistics. 15, 1, 152-162.

[15] Wahba, Grace. 1990. Spline Models for Observational Data. v. 59 in the CBMS-NSF Regional Conference Series in Applied Mathematics, SIAM, Philadelphia, PA, March 1990. Softcover, 169 pages, bibliography, author index. ISBN 0-89871-244-0

[16] Hall, Peter. 1983. "Large sample optimality of least squares cross-validation in density estimation." The Annals of Statistics. 11, 4, 1156-1174.

[17] Schuster, E.F., and G.G. Gregory. 1981. "On the nonconsistency of maximum likelihood nonparametric density estimators". Comp.Sci. & Statistics: Proc. 13th Symp. on the Interface. W.F. Eddy ed. 295-298. Springer-Verlag.

[18] Chow, Y.-S. and S.Geman, L.-D. Wu. 1983. "Consistent cross-validated density estimation." The Annals of Statistics. 11, 1, 25-38.

[19] Bowman, Adrian W. 1980. "A note on consistency of the kernel method for the analysis of categorical data." Biometrika (1980), 67, 3, pp. 682-4.

[20] Stone, Charles J. 1984 "An asymptotically optimal window selection rule for kernel density estimates." The Annals of Statistics. 12, 4, 1285-1297.

[21] Li, Ker-Chau. 1986. "Asymptotic optimality of $C_L$ and generalized cross-validation in ridge regression with application to spline smoothing." The Annals of Statistics. 14, 3, 1101-1112.

[22] Li, Ker-Chau. 1987. "Asymptotic optimality for $C_p$, $C_L$, cross-validation, and generalized cross-validation: discrete index set." The Annals of Statistics. 15, 3, 958-975.

[23] Utreras, Florencio I. 1987. "On generalized cross-validation for multivariate smoothing spline functions." SIAM J. Sci. Stat. Comput. 8, 4, July 1987.

[24] Andrews, Donald W.K. 1991. "Asymptotic optimality of generalized $C_L$, cross-validation, and generalized cross-validation in regression with heteroskedastic errors." Journal of Econometrics. 47 (1991) 359-377. North-Holland.

[25] Stone, M. 1977. "An asymptotic equivalence of choice of model by cross-validation and Akaike's criterion." J. Roy. Stat. Soc. Ser B, 39, 1, 44-47.

[26] Stone, M. 1977. "Asymptotics for and against cross-validation." Biometrika. 64, 1, 29-35.

[27] Billingsley, Patrick. 1986. Probability and Measure. Wiley, New York.

[28] Jennrich, R. 1969. "Asymptotic properties of nonlinear least squares estimators." Ann. Math. Stat. 40, 633-643.

[29] Liu, Yong. 1993. "Neural network model selection using asymptotic jackknife estimator and cross-validation method." In Giles, C.L., Hanson, S.J., and Cowan, J.D. (eds.), Advances in neural information processing systems 5, San Mateo, CA: Morgan Kaufmann Publishers.

[30] Moody, John E. 1992. "The effective number of parameters, an analysis of generalization and regularization in nonlinear learning system." In Moody, J.E., Hanson, S.J., and Lippmann, R.P., (eds.), Advances in neural information processing systems 4, San Mateo, CA: Morgan Kaufmann Publishers.

[31] Bailey, Timothy L. and Charles Elkan. 1993. "Estimating the accuracy of learned concepts." To appear in Proc. International Joint Conference on Artificial Intelligence.

[32] White, Halbert. 1993. Estimation, Inference, and Specification Analysis. Manuscript.

[33] White, Halbert. 1984. Asymptotic Theory for Econometricians. Academic Press.

[34] Klein, Erwin and Anthony C. Thompson. 1984 Theory of correspondences : including applications to mathematical economics. Wiley.